# DIGITAL REALISATION OF SELF-ORGANISING MAPS

Nigel M. Allinson       Martin J. Johnson       Kevin J. Moon

Department of Electronics
University of York
York
YO1 5DD
England

## ABSTRACT

A digital realisation of two-dimensional self-organising feature maps is presented. The method is based on subspace classification using an n-tuple technique. Weight vector approximation and orthogonal projections to produce a *winner-takes-all* network are also discussed. Over one million effective binary weights can be applied in 25ms using a conventional microcomputer. Details of a number of image recognition tasks, including character recognition and object centring, are described.

## INTRODUCTION

### Background

The overall aim of our work is to develop fast and flexible systems for image recognition, usually for commercial inspection tasks. There is an urgent need for automatic learning systems in such applications, since at present most systems employ heuristic classification techniques. This approach requires an extensive development effort for each new application, which exaggerates implementation costs; and for many tasks, there are no clearly defined features which can be employed for classification. Enquiring of a human expert will often only produce "good" and "bad" examples of each class and not the underlying strategies which he may employ. Our approach is to model in a quite abstract way the perceptual networks found in the mammalian brain for vision. A back-propagation network could be employed to generalise about the input pattern space, and it would find some useful representations. However, there are many difficulties with this approach, since the network structure assumes nothing about the input space and it can be difficult to bound complicated feature clusters using hyperplanes. The mammalian brain is a layered structure, and so another model may be proposed which involves the application of many two-dimensional feature maps. Each map takes information from the output of the preceding one and performs some type of clustering analysis in order to reduce the dimensionality of the input information. For successful recognition, similar patterns must be topologically close so that

novel patterns are in the same general area of the feature map as the class they are most like.    There is therefore a need for both global and local ordering processes within the feature map.  The process of global ordering in a topological map is termed, by Kohonen (1984), as *self-organisation.*
It is important to realize that all feedforward networks perform only one function, namely the labelling of areas in a pattern space.  This paper concentrates on a technique for realising large, fast, two-dimensional feature maps using a purely digital implementation.

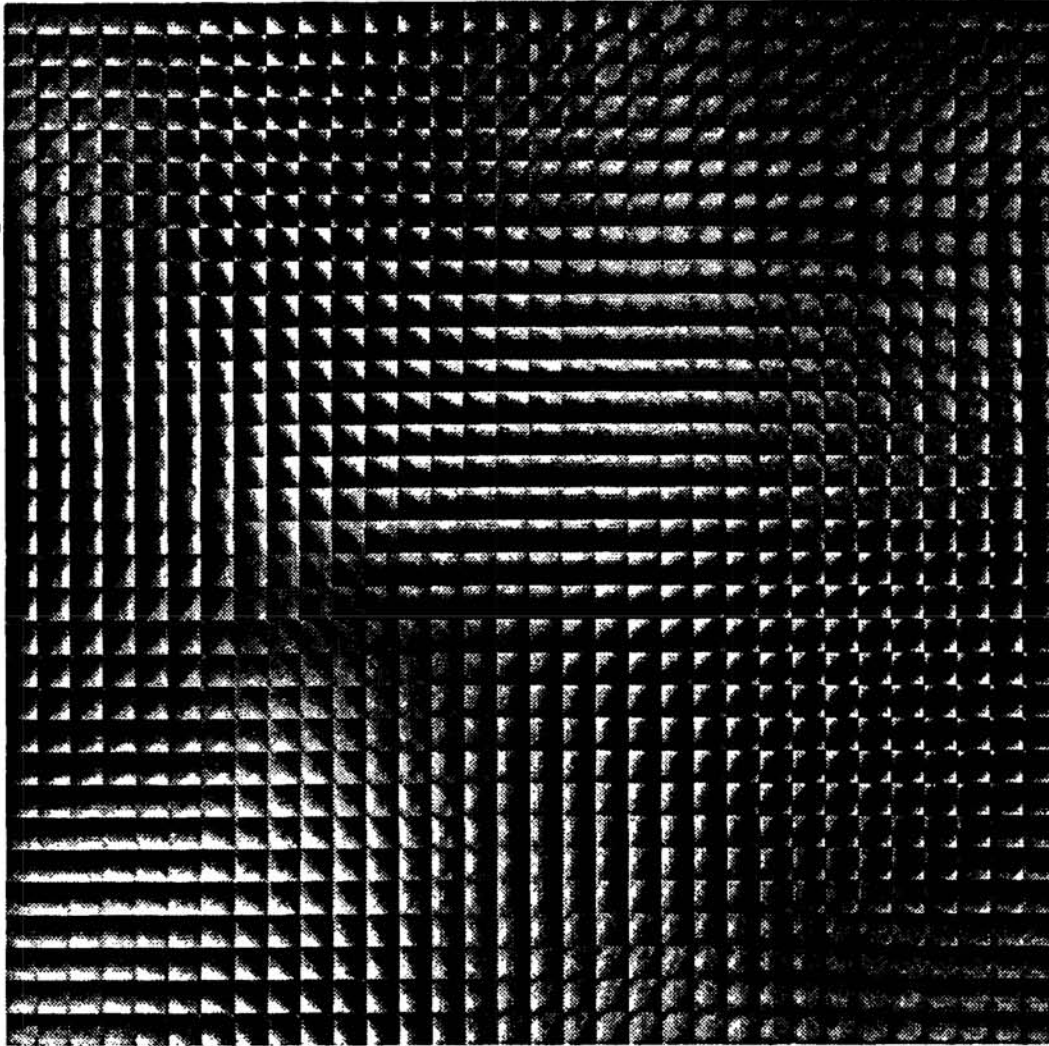

**Figure 1**. Unbounded Feature Map of Local Edges

## Self Organisation

Global ordering needs to adapt the entire neural map, but local ordering needs only local information.  Once the optimum global organisation has been found, then only more localised ordering can improve the topological organisation.  This process is the basis of the Kohonen clustering algorithm, where the specified area

of adaption decreases with time to give an increasing local ordering. It has been shown that this approach gives optimal ordering at global and local levels (Oja, 1983). It may be considered as a dimensionality reduction algorithm, and can be used as a vector quantiser.

Although Kohonen's self-organising feature maps have been successfully applied to speech recognition (Kohonen, 1988; Tattersall et al., 1988), there has been little investigation in their application for image recognition. Such feature maps can be used to extract various image primitives, such as textures, localised edges and terminations, at various scales of representations (Johnson and Allinson, 1988).

As a simple example, a test image of concentric circles is employed to construct a small feature map of localised edges (Figure 1). The distance measure used is the normalised dot product since in general magnitude information is unimportant. Under these conditions, each neuron output can be considered a similarity measure of the directions between the input pattern and the synaptic weight vector. This map shows that similar edges have been grouped together and that inverses are as far from each other as possible.

## DIGITAL IMPLEMENTATION

### Sub-Space Classification

Although a conventional serial computer is normally thought of as only performing one operation at a time, there is a task which it can successfully perform involving parallel computation. The action of addressing memory can be thought of as a highly parallel process, since it involves the comparison of a word, W, with a set of $2^N$ others where N is the number of bits in W. It is, in effect, performing $2^N$ parallel computations - each being a single match. This can be exploited to speed up the simulation of a network by using a conversion between conventional pattern space labelling and binary addressing.

Figure 2 shows how the labelling of two-dimensional pattern space is equivalent to the partitioning of the same space by the decision regions of a multiple layer perceptron. If each quantised part of the space is labelled with a number for each class then all that is necessary is for the pattern to be used as an address to give the stored label (i.e. the response) for each class. These labels may form a cluster of any shape and so multiple layers are not required to combine regions.
The apparent flaw in the above suggestion is that for anything other than a trivial problem, the labelling of every part of pattern space is impractical. For example a 32 x 32 input vector would require a memory of $2^{1024}$ words per unit! What is needed is a coding system which uses some basic assumptions about patterns in order to reduce the memory requirements. One assumption which can be made is that patterns will cluster together into various classes. As early as 1959, a method known as the n-tuple technique was used for pattern recognition (Bledsoe and Browning, 1959). This technique takes a number of subspaces of the pattern

# PERCEPTRON

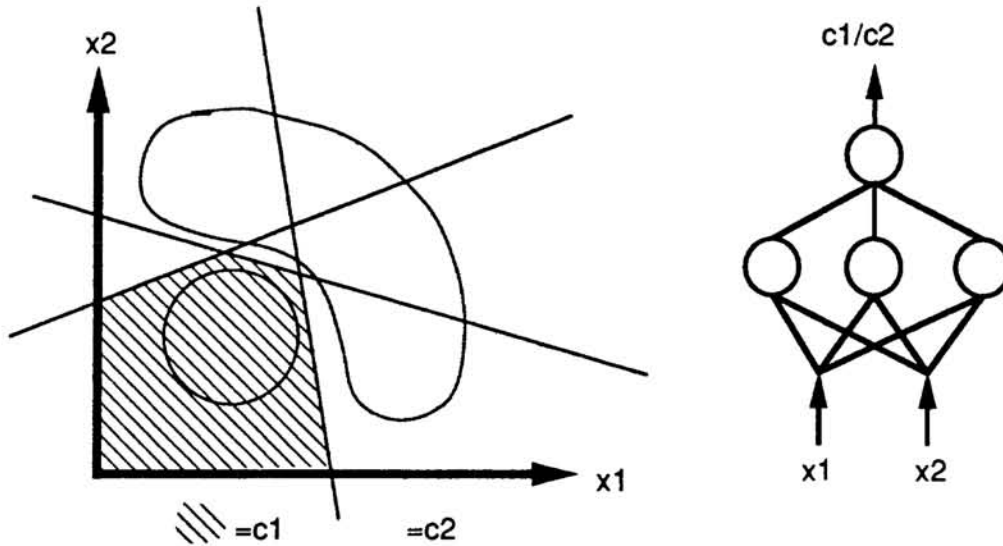

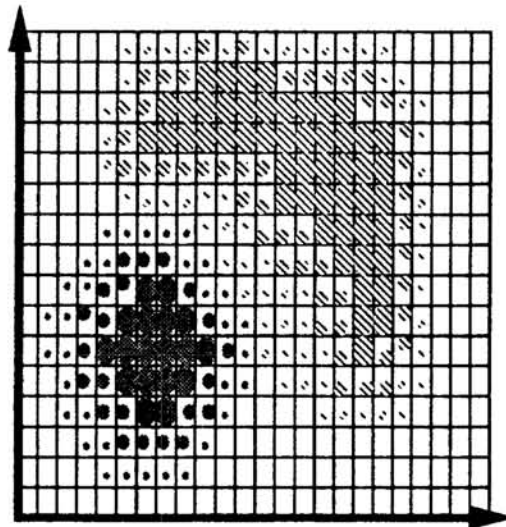

● = Class 1   ○ = Class 2

## LABELING

The labeling of a quantized subspace is equivalent to the partitioning of pattern space by the multi-layer perceptron.

**Figure 2.** Comparison of Perceptron and Sub-Space Classification

space and uses the sum of the resultant labels as the overall response. This gives a set of much smaller memories and inherent in the coding method is that similar patterns will have identical labels.

For example, assume a 16 bit pattern - 0101101001010100. Taking a four-bit sample from this, say bits 0-3, giving 0100. This can be used to address a 16 word memory to produce a single bit. If this bit is set to 1, then it is in effect labelling all patterns with 0100 as their first four bits; that is 4096 patterns of the form xxxxxxxxxxxx0100. Taking a second sample, namely bits 4-7 (0101). This labels xxxxxxxx0101xxxx patterns, but when added to the first sample there will be 256 patterns labelled twice (namely, xxxxxxxx01010100) and 7936 (i.e. 8192-256) labelled once. The third four-bit sample produces 16 patterns (namely,

xxxx101001010100) labelled three times. The fourth sample produces only one pattern 0101101001010100, which has been labelled four times. If an input pattern is applied which differs from this by one bit, then this will now be labelled three times by the samples; if it differs by two bits, it will either be labelled two or three times depending on whether the changes were in the same four-bit sample or not. Thus a distance measure is implicit in the coding method and reflects the assumed clustering of patterns. Applying this approach to the earlier problem of a 32 x 32 binary input vector and taking 128 eight-bit samples results in a distance measure between 0 and 128 and uses 32K bits of memory per unit.

**Weight Vector Approximation**

It is possible to make an estimate of the approximate weight vector for a particular sample from the bit table. For simplicity, consider a binary image from which t samples are taken to form a word, w, where

$$w = x_0 + 2x_1 + .... + 2^{t-1} x_{t-1}$$

This word can be used to address a vector W. Every bit in W[b] which is 1 either increases the weight vector probability where the respective bit in the address is set, or decreases if it is clear. Hence, if BIT [w,i] is the ith bit of w and A[i] is the contents of the memory {0, 1} then,

$$W[b] = \sum_{i=0}^{2^{t-1}} A[i] (2 \, BIT(b,i) - 1)$$

This represents an approximate measure of the weight element. Table 1 demonstrates the principle for a four-bit sample memory. Given randomly distributed inputs this binary vector is equivalent to the weight vector [2, 4, 0, -2].

If there is a large number of set bits in the memory for a particular unit then that will always give a high response - that is, it will become saturated. However, if there are too few bits set, this unit will not respond strongly to a general set of patterns. The number of bits must, therefore, be fixed at the start of training, distributed randomly within the memory and only redistribution of these bits allowed. Set bits could be taken from any other sample, but some samples will be more important than others. The proportion of 1's in an image should not be used as a measure, otherwise large uniform regions will be more significant than the pattern detail. This is a form of magnitude independent operation similar to the use of the normalised dot product applied in the analogue approach and so bits may only be moved from addresses with the same number of set bits as the current address.

**TABLE 1**. Weight Vector Approximation

| $x_3$ | $x_2$ | $x_1$ | $x_0$ | A | $W_3$ | $W_2$ | $W_1$ | $W_0$ | $x_3$ | $x_2$ | $x_1$ | $x_0$ | A | $W_3$ | $W_2$ | $W_1$ | $W_0$ |
|---|---|---|---|---|---|---|---|---|---|---|---|---|---|---|---|---|---|
| \multicolumn{5}{}{Address} | \multicolumn{4}{}{Weight change} | \multicolumn{5}{}{Address} | \multicolumn{4}{}{Weight change} |

| Address |  |  |  |  | Weight change |  |  |  | Address |  |  |  |  | Weight change |  |  |  |
|---|---|---|---|---|---|---|---|---|---|---|---|---|---|---|---|---|---|
| $x_3$ | $x_2$ | $x_1$ | $x_0$ | A | $W_3$ | $W_2$ | $W_1$ | $W_0$ | $x_3$ | $x_2$ | $x_1$ | $x_0$ | A | $W_3$ | $W_2$ | $W_1$ | $W_0$ |
| 0 | 0 | 0 | 0 | 0 |   |   |   |   | 1 | 0 | 0 | 0 | 1 | + | - | - | - |
| 0 | 0 | 0 | 1 | 0 |   |   |   |   | 1 | 0 | 0 | 1 | 0 |   |   |   |   |
| 0 | 0 | 1 | 0 | 0 |   |   |   |   | 1 | 0 | 1 | 0 | 0 |   |   |   |   |
| 0 | 0 | 1 | 1 | 1 | - | - | + | + | 1 | 0 | 1 | 1 | 0 |   |   |   |   |
| 0 | 1 | 0 | 0 | 1 | - | + | - | - | 1 | 1 | 0 | 0 | 1 | + | + | - | - |
| 0 | 1 | 0 | 1 | 0 |   |   |   |   | 1 | 1 | 0 | 1 | 1 | + | + | - | + |
| 0 | 1 | 1 | 0 | 1 | - | + | + | - | 1 | 1 | 1 | 0 | 1 | + | + | + | - |
| 0 | 1 | 1 | 1 | 0 |   |   |   |   | 1 | 1 | 1 | 1 | 1 | + | + | + | + |

|  |  |  |  |  |  |  |  |  | Equivalent weight vector |  |  |  |  | 2 | 4 | 0 | -2 |

## Orthogonal Projections

In order to speed up the simulation further, instead of representing each unit by a single bit in memory, each unit can be represented by a combination of bits. Hence many calculations can be effectively computed in parallel. The number of units which require a 1 for a particular sample will always be relatively small, and hence these can be coded. The coding method employed is to split the binary word, W, into x and y fields. These projection fields address a two dimensional map and so provide a fast technique of approximating the true content of the memory. The x bits are summed separately to the y bits, and together they give a good estimate of the unit co-ordinates with the most bits set in x and in y. This map becomes, in effect, a *winner-takes-all* network. The reducing neighbourhood of adaption employed in the Kohonen algorithm can also be readily incorporated by applying an overall mask to this map during the training phase.

Though only this output map is required during normal application of the system to image recognition tasks, it is possible to reconstruct the distribution of the two-dimensional weight vectors. Figure 3, using the technique illustrated in Table 1, shows this weight vector map for the concentric circle test image applied

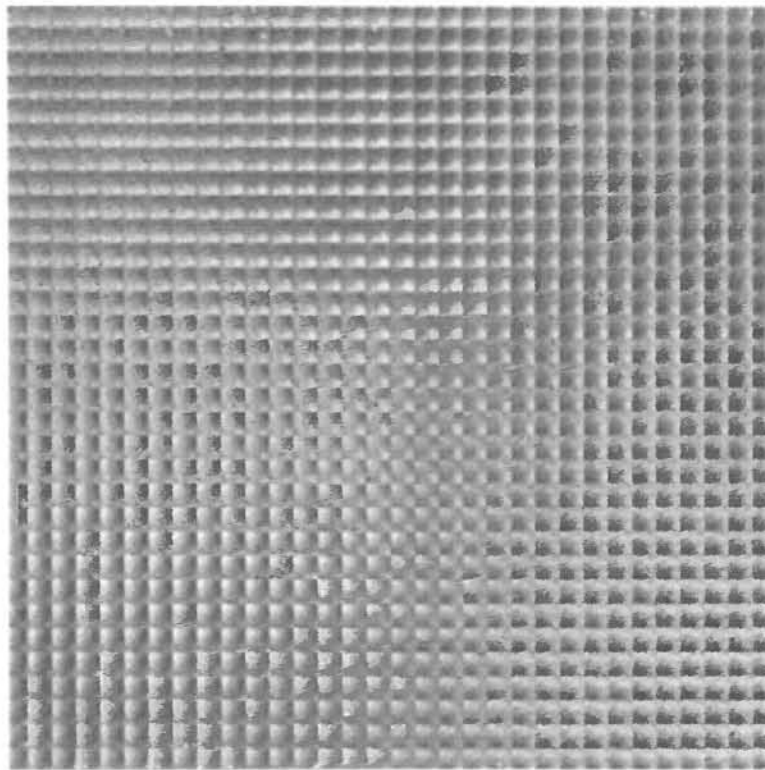

**Figure 3.** Reconstructed Feature Map of Local Edges

previously in the conventional analogue approach. This is a small digitised map containing 32 x 32 elements each with 16 x 16 input units and can be applied, using a general purpose desktop microcomputer running at 4 mips, in a few milliseconds.

## APPLICATION EXAMPLES

### Character Recognition

Though a long term objective remains the development of general purpose computer vision systems, with many layers of interacting feature maps together with suitable pre- and post-processing, many commercial tasks require decisions based on a constricted range of objects - that is their perceptual set is severely limited. However, ease of training and speed of application are paramount. An example of such an application involves the recognition of characters.

Figures 4 and 5 show an input pattern of hand-drawn A's and B's. The network, using the above digital technique, was given no information concerning the input image and the input window of 32 x 32 pixels was placed randomly on the image. The network took less than one minute to adapt and can be applied in 25 ms. This network is a 32 x 32 feature map of 32 x 32 elements, thus giving over one million effective weights. The output map forms two distinct clusters, one for A's in the top right corner of the map (Figure 4), and one for B's in the bottom left corner (Figure 5). If further characters are introduced in the input image then the output map will, during the training phase, self-organise to incorporate them.

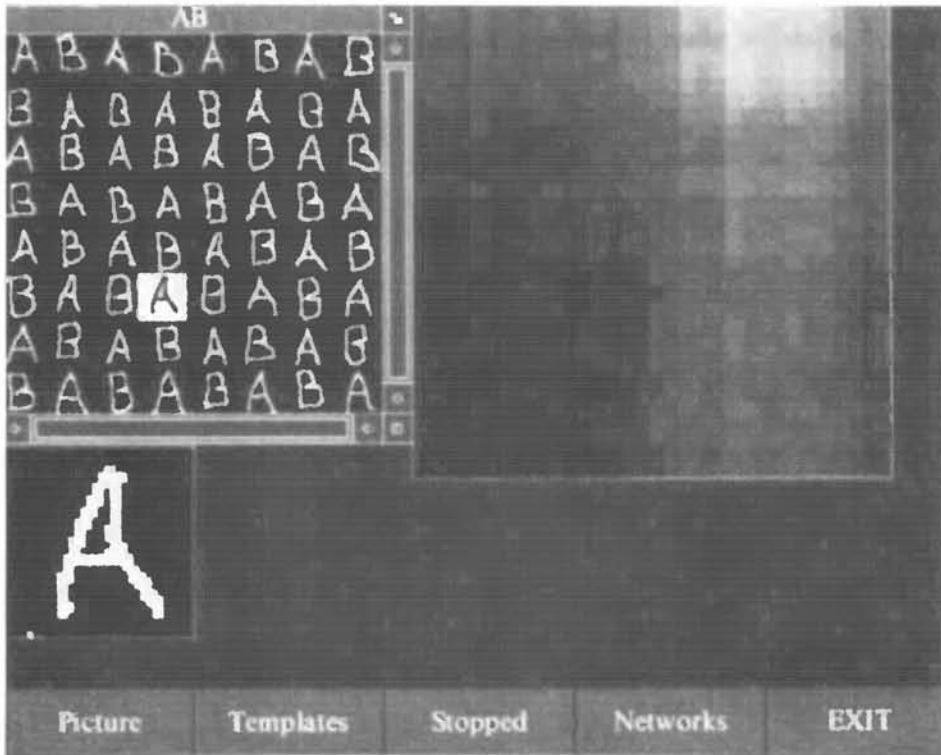

**Figure 4.** Trained Network Response for 'A' in Input Window

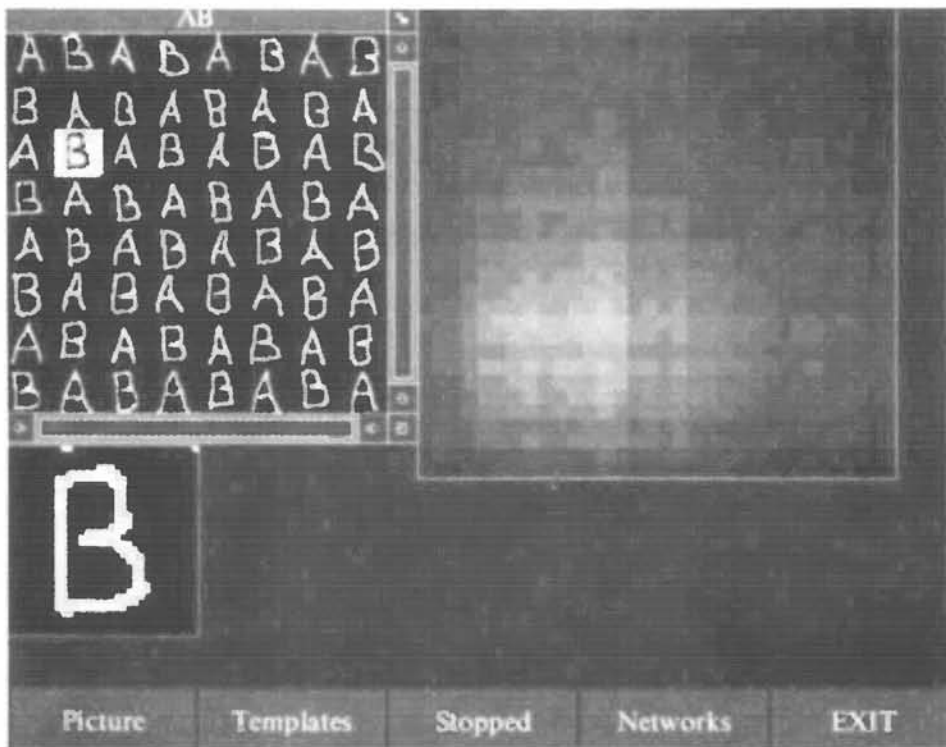

**Figure 5.** Trained Network Response for 'B' in Input Window

## Corrupted Images

Once the maximum response from the map is known, then the parts of the input window which caused it can be reconstructed to provide a form of *ideal* input pattern. The reconstructed input pattern is shown in the figures beneath the input image. This reconstruction can be employed to recognise occuluded patterns or to eliminate noise in subsequent input images.

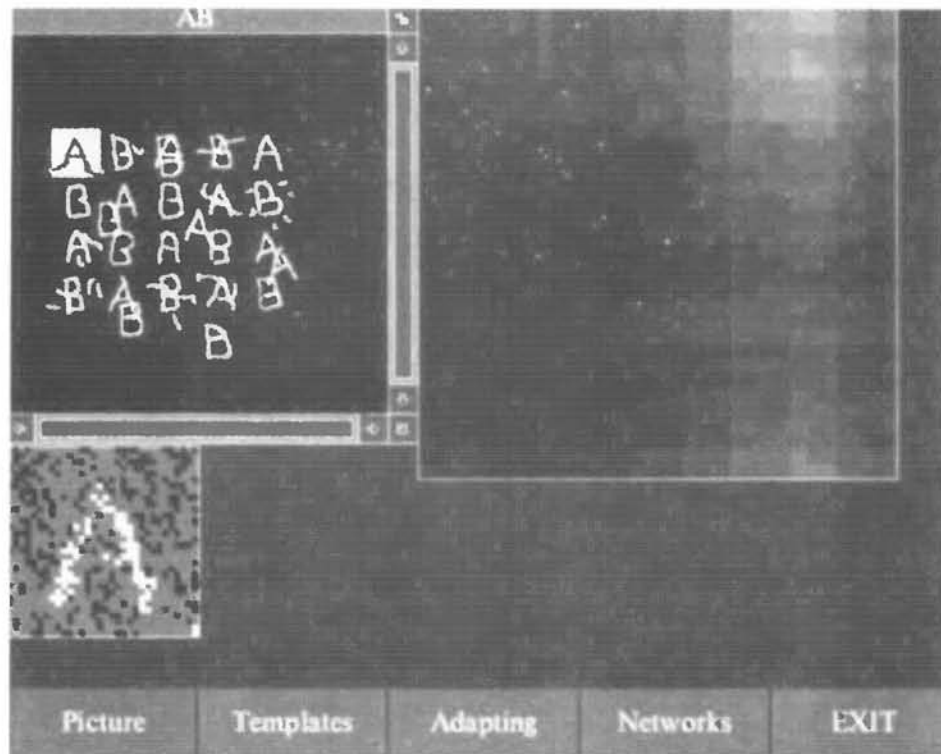

**Figure 6**. Trained Network Response for Corrupted 'A' in Input Window. Reconstructed Input Pattern Shown Below Test Image

Figure 6 shows the response of the network, trained on the input image of Figures 4 and 5, to a corrupted image of A's and B's. It has still managed to recognise the input character as an A, but the reconstructed version shows that the *extra noise* has been eliminated.

## Object Centring

The centering of an object within the input window permits the application of conformant mapping strategies, such as polar exponential grids, to be applied which yields scale and rotation invariant recognition. The same network as employed in the previous example was used, but a target position for the maximum network response was specified and the network was adapted half-way between this and the actual maximum response location.

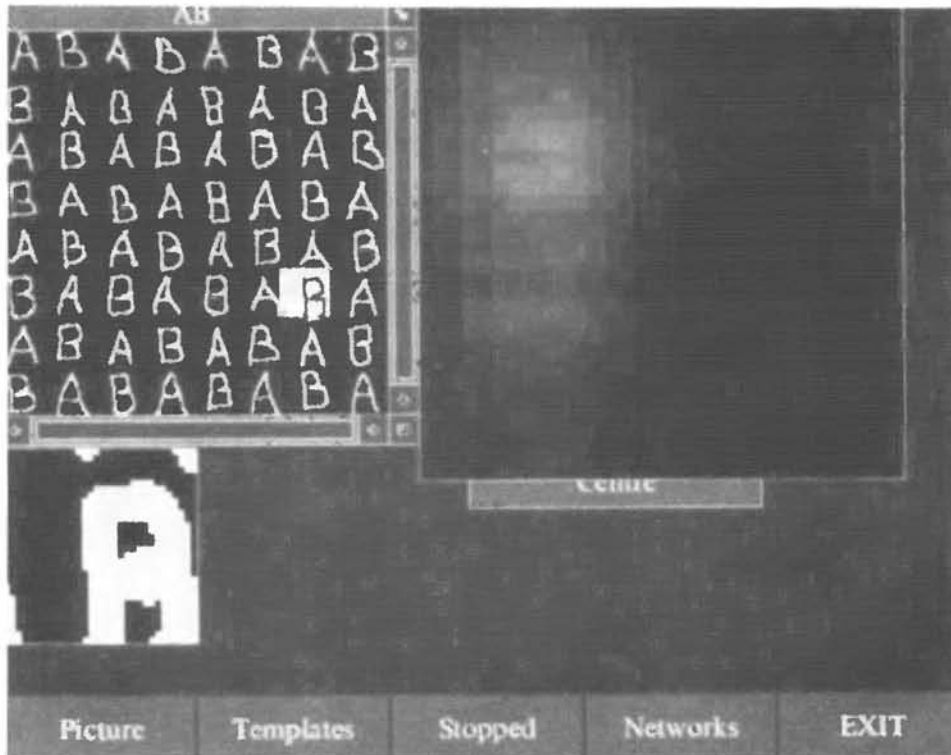

**Figure 7**. Trained Network Response for Off-Centred Character. Input Window is Low-Pass Filtered as shown.

Figure 7 shows such a network. When the response is in the centre of the output map then an input object (character) is centred in the recognition window. In the example shown, there is an off-centred response of the trained network for an off-centred character. This deviation is used to change the position of the input window. Once centering has been achieved, object recognition can occur.

## CONCLUSIONS

The application of unsupervised feature maps for image recognition has been demonstrated. The digital realisation technique permits the application of large maps, which can be applied in *real time* using conventional microcomputers. The use of orthogonal projections to give a *winner-take-all network* reduces memory requirements by approximately 30-fold and gives a computational cost of $0(n^{1/2})$, where n is the number of elements in the map. The general approach can be applied in any form of feedforward neural network.

### Acknowledgements

This work has been supported by the Innovation and Research Priming Fund of the University of York.

**References**

W. W. Bledsoe and I. Browning.  Pattern Recognition and Reading by Machine. *Proc. East. Joint Comp. Conf.*, 225-232 (1959).

M. J. Johnson and N. M. Allinson.  An Advanced Neural Network for Visual Pattern Recognition. *Proc. UKIT 88*, Swansea, 296-299 (1988).

T. Kohonen.  Self Organization and Associative Memory.  Springer-Verlag, Berlin (1984).

T. Kohonen.  The 'Neural' Phonetic Typewriter. *Computer* 21,11-22 (1988).

E. Oja.  Subspace Methods of Pattern Recognition. *Research Studies Press*, Letchworth (1983).

G. D. Tattersall, P. W. Linford and R. Linggard.  Neural Arrays for Speech Recognition. *Br. Telecom Technol. J.* 6, 140-163 (1988).
